# Video Annotation and Tracking with Active Learning

**Carl Vondrick**
UC Irvine
vondrick@mit.edu

**Deva Ramanan**
UC Irvine
dramanan@ics.uci.edu

## Abstract

We introduce a novel active learning framework for video annotation. By judiciously choosing which frames a user should annotate, we can obtain highly accurate tracks with minimal user effort. We cast this problem as one of active learning, and show that we can obtain excellent performance by querying frames that, if annotated, would produce a large expected change in the estimated object track. We implement a constrained tracker and compute the expected change for putative annotations with efficient dynamic programming algorithms. We demonstrate our framework on four datasets, including two benchmark datasets constructed with key frame annotations obtained by Amazon Mechanical Turk. Our results indicate that we could obtain equivalent labels for a small fraction of the original cost.

## 1   Introduction

With the decreasing costs of personal portable cameras and the rise of online video sharing services such as YouTube, there is an abundance of *unlabeled* video readily available. To both train and evaluate computer vision models for video analysis, this data must be labeled. Indeed, many approaches have demonstrated the power of data-driven analysis given *labeled* video footage [12, 17].

But, annotating massive videos is prohibitively expensive. The twenty-six hour VIRAT video data set consisting of surveillance footage of cars and people cost *tens of thousands of dollars* to annotate despite deploying state-of-the-art annotation protocols [13]. Existing video annotation protocols typically work by having users (possibly on Amazon Mechanical Turk) label a sparse set of key frames followed by either linear interpolation [16] or nonlinear tracking [1, 15].

We propose an adaptive key-frame strategy which uses active learning to *intelligently* query a worker to label only certain objects at only certain frames that are likely to improve performance. This approach exploits the fact, that for real footage, not all objects/frames are "created equal"; some objects during some frames are "easy" to automatically annotate in that they are stationary (such as parked cars in VIRAT [13]) or moving in isolation (such a single basketball player running down the court during a fast break [15]). In these cases, a few user clicks are enough to constrain a visual tracker to produce accurate tracks. Rather, user clicks should be spent on more "hard" objects/frames that are visually ambiguous, such as occlusions or cluttered backgrounds.

**Related work (Active learning):** We refer the reader to the excellent survey in [14] for a contemporary review of active learning. Our approach is an instance of active structured prediction

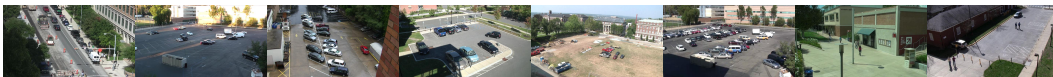

Figure 1: Videos from the VIRAT data set [13] can have hundreds of objects per frame. Many of those objects are easily tracked except for a few difficult cases. Our active learning framework automatically focuses the worker's effort on the difficult instances (such as occlusion or deformation).

[8, 7], since we train object models that predict a complex, structured label (an object track) rather than a binary class output. However, rather than training a single car model over several videos (which must be invariant to instance-specific properties such as color and shape), we train a separate car model for each car instance to be tracked. From this perspective, our training examples are individual frames rather than videos. But notably, these examples are *non-i.i.d*; indeed, temporal dependencies are crucial for obtaining tracks from sparse labels. We believe this property makes video a prime candidate for active learning, possibly simplifying its theoretical analysis [14, 2] because one does not face an adversarial ordering of data. Our approach is similar to recent work in active labeling [4], except we determine which part of the label the user should annotate in order to improve performance the most. Finally, we use a novel query strategy appropriate for video: rather than use expected information gain (expensive to compute for structured predictors) or label entropy (too coarse of an approximation), we use the expected label change to select a frame. We select the frame, that when labeled, will produce the largest change in the estimated track of an object.

**Related work (Interactive video annotation):** There has also been work on interactive tracking from the computer vision community. [5] describe efficient data structures that enable interactive tracking, but do not focus on frame query strategies as we do. [16] and [1] describe systems that allow users to manually correct drifting trackers, but this requires annotators to watch an entire video in order to determine such erroneous frames, a significant burden in our experience.

## 2 Tracking

In this section, we outline the dynamic programming tracker of [15]. We will extend it in Section 3 to construct an efficient active learning algorithm. We begin by describing a method for tracking a single object, given a sparse set of key frame bounding-box annotations. As in [15], we use a visual tracker to interpolate the annotations for the unlabeled in-between frames. We define $b_t^i$ to be a bounding box at frame $t$ at pixel position $i$. Let $\zeta$ be the non-empty set of worker annotations, represented as a set of bounding boxes. Without loss of generality, assume that all paths are on the interval $0 \leq t \leq T$.

### 2.1 Discriminative Object Templates

We build a discriminative visual model of the object in order to predict its location. For every bounding box annotation in $\zeta$, we extract its associated image patch and resize it to the average size in the set. We then extract both histogram of oriented gradients (HOG) [9] and color features: $\phi_n(b_n) = [HOG \quad RGB]^T$ where RGB are the means and covariances of the color channels. When trained with a linear classifier, these color features are able to learn a quadratic decision boundary in RGB-space. In our experiments, we used a HOG bin size of either 4 or 8 depending on the size of the object.

We then learn a model trained to discriminate the object against the the background. For every annotated frame, we extract an *extremely* large set of negative bounding boxes that do not significantly overlap with the positive instances. Given a set of features $b_n$ with labels $y_n \in \{-1, 1\}$ classifying them as positive or negative, we train a linear SVM by minimizing the loss function:

$$w^* = \operatorname{argmin} \frac{1}{2} w \cdot w + C \sum_n^N \max(0, 1 - y_n w \cdot \phi_n(b_n)) \tag{1}$$

We use liblinear [10] in our experiments. Training typically took only a few seconds.

### 2.2 Motion Model

In order to score a putative interpolated path $b_{0:T} = \{b_0 \dots b_T\}$, we define the energy function $E(b_{0:T})$ comprised of both unary and pairwise terms:

$$E(b_{0:T}) = \sum_{t=0}^{T} U_t(b_t) + S(b_t, b_{t-1}) \tag{2}$$

$$U_t(b_t) = \min\left(-w \cdot \phi_t(b_t), \alpha_1\right), \quad S(b_t, b_{t-1}) = \alpha_2 ||b_t - b_{t-1}||^2 \tag{3}$$

where $U_t(b_t)$ is the local match cost and $S_t(b_t, b_{t-1})$ is the pairwise spring. $U_t(b_t)$ scores how well a particular $b_t$ matches against the learned appearance model $w$, but truncated by $\alpha_1$ so as to reduce the penalty when the object undergoes an occlusion. We are able to efficiently compute the dot product $w \cdot \phi_t(b_t)$ using integral images on the RGB weights [6]. $S_t(b_t, b_{t-1})$ favors smooth motion and prevents the tracked object from teleporting across the scene.

## 2.3   Efficient Optimization

We can recover the missing annotations by computing the optimal path as given by the energy function. We find the least cost path $b^*_{0:T}$ over the exponential set of all possible paths:

$$b^*_{0:T} = \operatorname*{argmin}_{b_{0:T}} E(b_{0:T}) \quad s.t. \quad b_t = b^i_t \quad \forall b^i_t \in \zeta \tag{4}$$

subject to the constraint that the path crosses through the annotations labeled by the worker in $\zeta$. We note that these constraints can be removed by simply redefining $U_t(b_t) = \infty \quad \forall b_t \neq b^i_t$.

A naive approach to minimizing (4) would take $O(K^T)$ for $K$ locations per frame. However, we can efficiently solve the above problem in $O(TK^2)$ by using dynamic programming through a forward pass recursion [3]:

$$C^{\rightarrow}_0(b_0) = U_0(b_0)$$
$$C^{\rightarrow}_t(b_t) = U_t(b_t) + \min_{b_{t-1}} C^{\rightarrow}_{t-1}(b_{t-1}) + S(b_t, b_{t-1}) \tag{5}$$

$$\pi^{\rightarrow}_t(b_t) = \operatorname*{argmin}_{b_{t-1}} C^{\rightarrow}_{t-1}(b_{t-1}) + S(b_t, b_{t-1}) \tag{6}$$

By storing the pointers in (6), we are able to reconstruct the least cost path by backtracking from the last frame $T$. We note that we can further reduce this computation to $O(TK)$ by applying distance transform speed ups to the pairwise term in (3) [11].

# 3   Active Learning

Let curr$_{0:T}$ be the current best estimate for the path given a set of user annotations $\zeta$. We wish to compute which frame the user should annotate next $t^*$. In the ideal case, if we had knowledge of the ground-truth path $b^{gt}_{0:T}$, we should select the frame $t$, that when annotated with $b^{gt}_t$, would produce a new estimated path closest to the ground-truth. Let us write next$_{0:T}(b^{gt}_t)$ for the estimated track given the augmented constraint set $\zeta' = \zeta \cup b^{gt}_t$. The optimal next frame is:

$$t^{opt} = \operatorname*{argmin}_{0 \leq t \leq T} \sum_{j=0}^{T} \operatorname{err}(b^{gt}_j, \operatorname{next}_j(b^{gt}_t)) \tag{7}$$

where $err$ could be squared error or a thresholded overlap (in which $err$ evaluates to 0 or 1 depending upon if the two locations sufficiently overlap or not). Unfortunately, we cannot directly compute (7) since we do not know the true labels ahead of time.

## 3.1   Maximum Expected Label Change (ELC)

We make two simplifying assumptions to implement the previous ideal selection strategy, inspired by the popular maximum expected gradient length (EGL) algorithm for active learning [14] (which selects an example so as to maximize the expected change in a learned model). First, we change the minimization to a maximization and replace the ground-truth error with the change in track label: $err(b^{gt}_j, \operatorname{next}_j(b^{gt}_t)) \Rightarrow err(\operatorname{curr}_j, \operatorname{next}_j(b^{gt}_t))$. Intuitively, if we make a large change in the estimated track, we are likely to be taking a large step toward the ground-truth solution. However, this requires knowing the ground-truth location $b^{gt}_t$. We make the second assumption that we have access to an accurate estimate of $P(b^i_t)$, which is the probability that, if we show the user frame $t$, then they will annotate a particular location $i$. We can use this distribution to compute an expected change in track label:

$$t^* = \operatorname*{argmax}_{0 \leq t \leq T} \sum_{i=0}^{K} P(b^i_t) \cdot \Delta I(b^i_t) \quad \text{where} \quad \Delta I(b^i_t) = \sum_{j=0}^{T} \operatorname{err}(\operatorname{curr}_j, \operatorname{next}_j(b^i_t)) \tag{8}$$

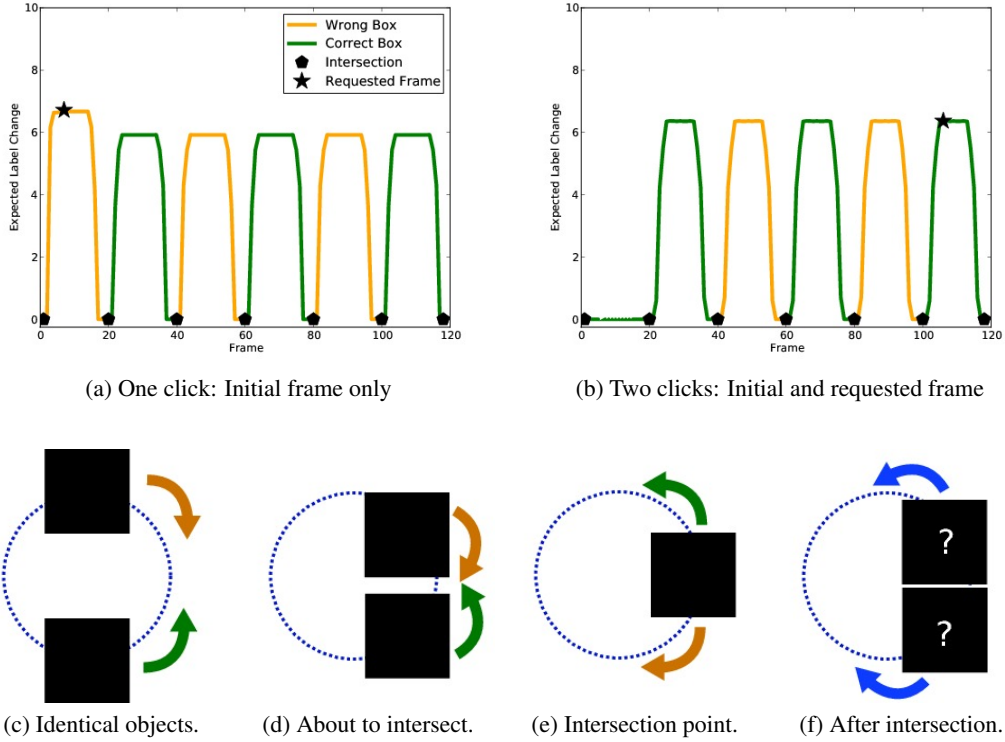

(a) One click: Initial frame only    (b) Two clicks: Initial and requested frame

(c) Identical objects.    (d) About to intersect.    (e) Intersection point.    (f) After intersection.

Figure 2: We consider a synthetic video of two nearly identical rectangles rotating around a point—one clockwise and the other counterclockwise. The rectangles intersect every 20 frames, at which point the tracker does not know which direction the true rectangle is following. Did they bounce or pass through? (**a**) Our framework realizes the ambiguity can be resolved by requesting annotations when they do not intersect. Due to the periodic motion, a fixed rate tracker may request annotations at the intersection points, resulting in wasted clicks. The expected label change plateaus because every point along the maximas provide the same amount of disambiguating information. (**b**) Once the requested frame is annotated, that corresponding segment is resolved, but the others remain ambiguous. In this example, our framework can determine the true path for a particular rectangle in only 7 clicks, while a fixed rate tracker may require 13 clicks.

The above selects the frame, that when annotated, produces the largest expected track label change. We now show how to compute $P(b_t^i)$ and $\Delta I(b_t^i)$ using costs and constrained paths, respectively, from the dynamic-programming based visual tracker described in Section 2. By considering every possible space-time location that a worker could annotate, we are able to determine which frame we expect could change the current path the most. Even though this calculation searches over an exponential number of paths, we are able to compute it in polynomial time using dynamic programming. Moreover, (8) can be parallelized across frames in order to guarantee a rapid response time, often necessary due to the interactive nature of active learning.

## 3.2 Annotation Likelihood and Estimated Tracks

A user has access to global knowledge and video history when annotating a frame. To capture such global information, we define the annotation likelihood of location $b_t^i$ to be the score of the best track given that additional annotation:

$$P(b_t^i) \propto \exp\left(\frac{-\Psi(b_t^i)}{\sigma^2}\right) \quad \text{where} \quad \Psi(b_t^i) = E\left(\text{next}_{0:T}(b_t^i)\right) \tag{9}$$

The above formulation only assigns high probabilities to locations that lie on paths that agree with the global constraints in $\zeta$, as explained in Fig.2 and Fig.3. To compute energies $\Psi(b_t^i)$ for all

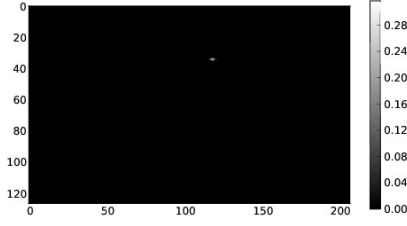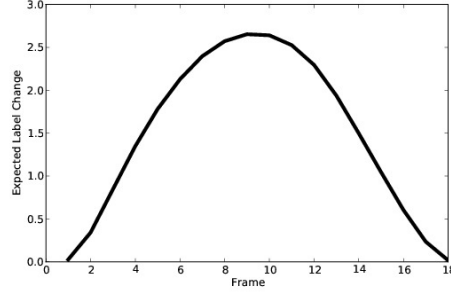

Figure 3: Consider two identical rectangles that translate, but never intersect. Although both objects have the same appearance, our framework does not query for new annotations because the pairwise cost has made it unlikely that the two objects switch identities, indicated by a single mode in the probability map. A probability exclusively using unary terms would be bimodal.

Figure 4: Consider a white rectangle moving on a white background. Since it is impossible to distuingish the foreground from the background, our framework will query for the midpoint and gracefully degrade to a fixed rate labeling. If the object is extremely difficult to localize, the active learner will automatically decide the optimal annotation strategy is to use fixed rate key frames.

spacetime locations $b_t^i$, we use a standard two-pass dynamic programming algorithm for computing *min-marginals*:

$$\Psi(b_t^i) = C_t^{\rightarrow}(b_t^i) + C_t^{\leftarrow}(b_t^i) - U(b_t^i) \tag{10}$$

where $C_t^{\leftarrow}(b_t^i)$ corresponds to intermediate costs computed by running the recursive algorithm from (5) backward in time. By caching forward and backward pointers $\pi_t^{\rightarrow}(b_t^i)$ and $\pi_t^{\leftarrow}(b_t^i)$, the associated tracks $\text{next}_{0:T}(b_t^i)$ can be found by backtracking both forward and backward from any spacetime location $b_t^i$.

### 3.3 Label Change

We now describe a dynamic programming algorithm for computing the label change $\Delta I(b_t^i)$ for all possible spacetime locations $b_t^i$. To do so, we define intermediate quantities $\Theta_t^{\rightarrow}(b_t^i)$ which represent the label change up to time $t$ given the user annotates location $b_t^i$:

$$\Theta_0^{\rightarrow}(b_0) = \text{err}(\text{curr}_0, \text{next}_0(b_0)) \tag{11}$$
$$\Theta_t^{\rightarrow}(b_t) = \text{err}(\text{curr}_t, \text{next}_t(b_t)) + \Theta_{t-1}^{\rightarrow}(\pi_t^{\rightarrow}(b_t)) \tag{12}$$

We can compute $\Theta_t^{\leftarrow}(b_t^i)$, the expected label change due to frames $t$ to $T$ given a user annotation at $b_t^i$, by running the above recursion backward in time. The total label change is their sum, minus the double-counted error from frame $t$:

$$\Delta I(b_t^i) = \Theta_t^{\rightarrow}(b_t^i) + \Theta_t^{\leftarrow}(b_t^i) - \text{err}(\text{curr}_t, \text{next}_t(b_t^i)) \tag{13}$$

(13) is sensitive to small spatial shifts; i.e. $\Delta I(b_t^i) \not\approx \Delta I(b_t^{i+\epsilon})$. To reduce the effect of imprecise human labeling (which we encounter in practice), we replace the label change with a worst-case label change computed over a neighboring window $N(b_t^i)$:

$$\tilde{\Delta} I(b_t^i) = \min_{b_t^j \in N(b_t^i)} \Delta I(b_t^j) \tag{14}$$

By selecting frames that have a large expected "worse-case" label change, we avoid querying frames that require precise labeling and instead query for frames that are easy to label (e.g., the user may annotate any location within a small neighborhood and still produce a large label change).

### 3.4 Stopping Criteria

Our final active learning algorithm is as follows: we query a frame $t^*$ according to (8), add the user annotation to the constraint set $\zeta$, retrain the object template with additional training examples

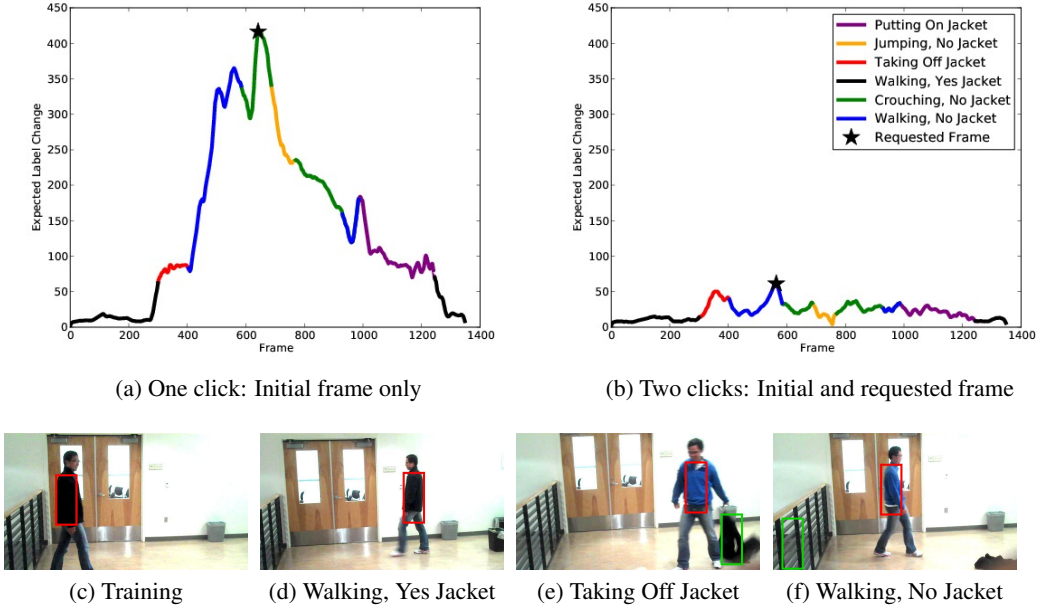

(a) One click: Initial frame only                  (b) Two clicks: Initial and requested frame

(c) Training          (d) Walking, Yes Jacket          (e) Taking Off Jacket          (f) Walking, No Jacket

Figure 5: We analyze a video of a man who takes off a jacket and changes his pose. A tracker trained only on the initial frame will lose the object when his appearance changes. Our framework is able to determine which additional frame the user should annotate in order to resolve the track. (**a**) Our framework does not expect any significant label change when the person is wearing the same jacket as in the training frame (black curve). But, when the jacket is removed and the person changes his pose (colorful curves), the tracker cannot localize the object and our framework queries for an additional annotation. (**b**) After annotating the requested frame, the tracker learns the color of the person's shirt and gains confidence in its track estimate. A fixed rate tracker may pick a frame where the person is still wearing the jacket, resulting in a wasted click. (**c-f**) The green box is the predicted path with one click and red box is with two clicks. If there is no green box, it is the same as the red.

extracted from frame $t^*$ (according to (1)), and repeat. We stop requesting annotations once we are confident that additional annotations will not significantly change the predicted path:

$$\max_{0 \leq t \leq T} \sum_{i=0}^{K} P(b_t^i) \cdot \Delta I(b_t^i) < \text{tolerance} \qquad (15)$$

We then report $b_{0:T}^*$ as the final annotated track as found in (4). We note, however, that in practice external factors, such as budget, will often trigger the stopping condition before we have obtained a perfect track. As long as the budget is sufficiently high, the reported annotations will closely match the actual location of the tracked object.

We also note that one can apply our active learning algorithm *in parallel* for multiple objects in a video. We maintain separate object models $w$ and constraint sets $\zeta$ for each object. We select the object and frame with the maximum expected label change according to (8). We demonstrate that this strategy naturally focuses labeling effort on the more difficult objects in a video.

## 4   Qualitative Experiments

In order to demonstrate our framework's capabilities, we show how our approach handles a couple of interesting annotation problems. We have assembled two data sets: a synthetic video of easy-to-localize rectangles maneuvering in an uncluttered background, and a real-world data set of actors following scripted walking patterns.

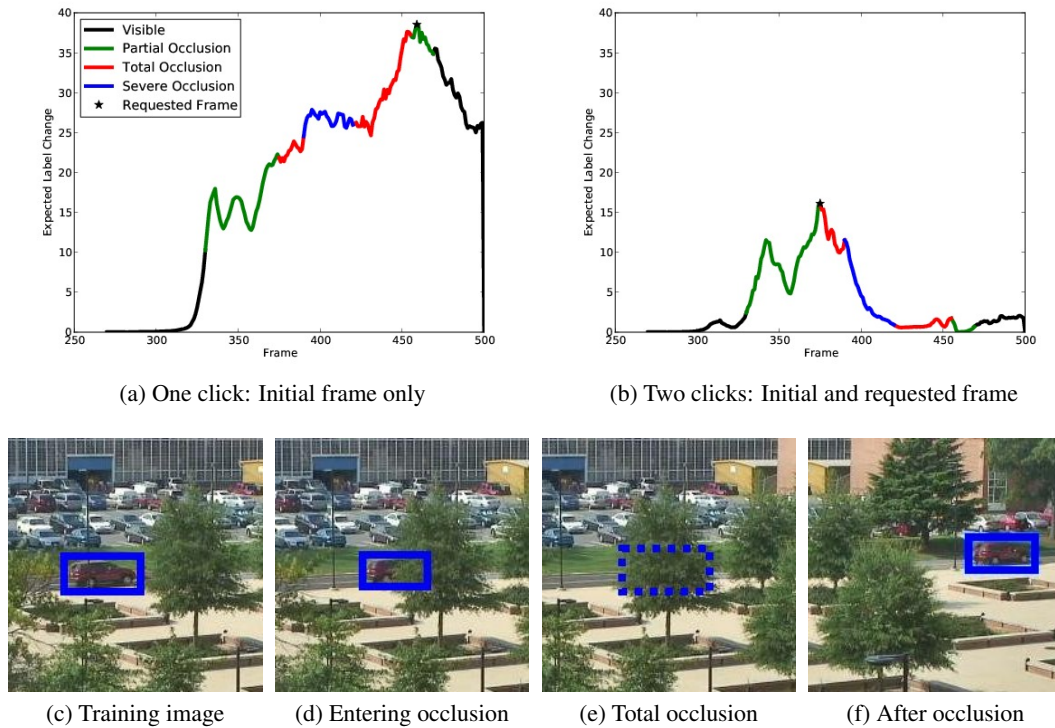

(a) One click: Initial frame only          (b) Two clicks: Initial and requested frame

(c) Training image    (d) Entering occlusion    (e) Total occlusion    (f) After occlusion

Figure 6: We investigate a car from [13] that undergoes a total occlusion and later reappears. The tracker is able to localize the car until it enters the occlusion, but it cannot recover when the car reappears. (**a**) Our framework expects a large label change during the occlusion and when the object is lost. The largest label change occurs when the object begins to reappear because this frame would lock the tracker back onto the correct path. (**b**) When the tracker receives the requested annotation, it is able to recover from the occlusion, but it is still confused when the object is not visible.

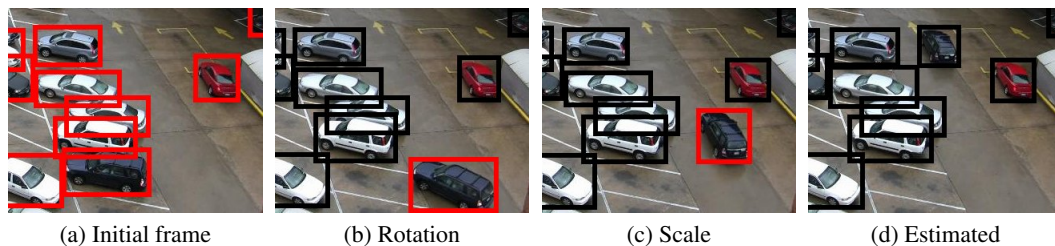

(a) Initial frame          (b) Rotation          (c) Scale          (d) Estimated

Figure 7: We examine situations where there are many easy-to-localize objects (e.g., stationary objects) and only a few difficult instances. In this example, red boxes were manually annotated and black boxes are automatically estimated. Our framework realizes that the stationary objects are not likely to change their label, so it focuses annotations on moving objects.

We refer the reader to the figures. Fig.2 shows how our framework is able to resolve inherently ambiguous motion with the minimum number of annotations. Fig.3 highlights how our framework does not request annotations when the paths of two identical objects are disjoint because the motion is not ambiguous. Fig.4 reveals how our framework will gracefully degrade to fixed rate key frames if the tracked object is difficult to localize. Fig.5 demonstrates motion of objects that deform. Fig.6 shows how we are able to detect occlusions and automatically recover by querying for a correct annotation. Finally, Fig.7 shows how we are able to transfer wasted clicks from stationary objects on to moving objects.

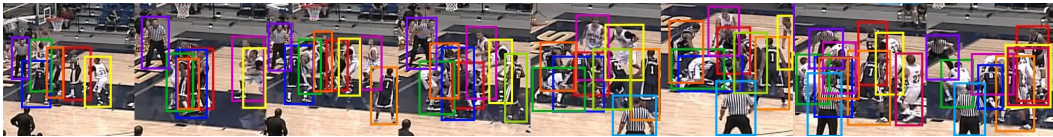

Figure 8: A hard scene in a basketball game [15]. Players frequently undergo total and partial occlusion, alter their pose, and are difficult to localize due to a cluttered background.

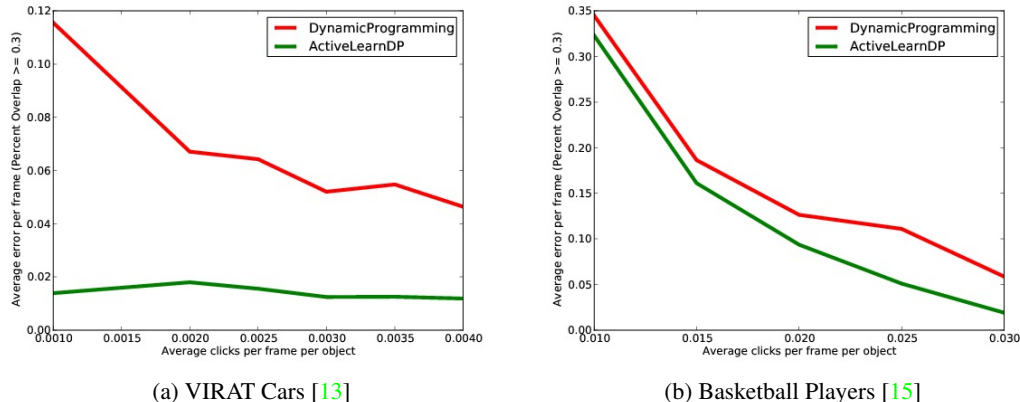

(a) VIRAT Cars [13]                                    (b) Basketball Players [15]

Figure 9: We compare active key frames (green curve) vs. fixed rate key frames (red curve) on a subset (a few thousand frames) of the VIRAT videos and part of a basketball game. We could improve performance by increasing annotation frequency, but this also increases the cost. By decreasing the annotation frequency in the easy sections and instead transferring those clicks to the difficult frames, we achieve superior performance over the current methods on the same budget. (**a**) Due to the large number of stationary objects in VIRAT, our framework assigns a tremendous number of clicks to moving objects, allowing us to achieve nearly zero error. (**b**) By focusing annotation effort on ambiguous frames, we show nearly a 5% improvement on basketball players.

## 5   Benchmark Results

We validate our approach on both the VIRAT challenge video surveillance data set [13] and the basketball game studied in [15]. VIRAT is unique for its enormous size of over three million frames and up to hundreds of annotated objects in each frame. The basketball game is extremely difficult due to cluttered backgrounds, motion blur, frequent occlusions, and drastic pose changes.

We evaluate the performance of our tracker using active key frames versus fixed rate key frames. A fixed rate tracker simply requests annotations every $T$ frames, regardless of the video content. For active key frames, we use the annotation schedule presented in section 3. Our key frame baseline is the state-of-the-art labeling protocol used to originally annotate both datasets [15, 13]. In a given video, we allow our active learning protocol to iteratively pick a frame and an object to annotate until the budget is exhausted. We then run the tracker described in section 2 constrained by these key frames and compare its performance.

We score the two key frame schedules by determining how well the tracker is able to estimate the ground truth annotations. For every frame, we consider a prediction to be correct as long as it overlaps the ground truth by at least 30%, a threshold that agrees with our qualitative rating of performance. We compare our active approach to a fixed-rate baseline for a fixed amount of user effort: is it better to spend $X$ user clicks on active or fixed-rate key frames? Fig.9 shows the former strategy is better. Indeed, we can annotate the VIRAT data set for *one tenth* of its original cost.

**Acknowledgements:**   Funding for this research was provided by NSF grants 0954083 and 0812428, ONR-MURI Grant N00014-10-1-0933, an NSF GRF, and support from Intel and Amazon.

# References

[1] A. Agarwala, A. Hertzmann, D. Salesin, and S. Seitz. Keyframe-based tracking for rotoscoping and animation. In *ACM Transactions on Graphics (TOG)*, volume 23, pages 584–591. ACM, 2004. 1, 2

[2] M.-F. Balcan, A. Beygelzimer, and J. Langford. Agnostic active learning. In *Proceedings of the 23rd international conference on Machine learning*, ICML '06, pages 65–72, New York, NY, USA, 2006. ACM. 2

[3] R. Bellman. Some problems in the theory of dynamic programming. *Econometrica: Journal of the Econometric Society*, pages 37–48, 1954. 3

[4] S. Branson, P. Perona, and S. Belongie. Strong supervision from weak annotation: Interactive training of deformable part models. *ICCV*. 2

[5] A. Buchanan and A. Fitzgibbon. Interactive feature tracking using kd trees and dynamic programming. In *CVPR 06*, volume 1, pages 626–633. Citeseer, 2006. 2

[6] F. Crow. Summed-area tables for texture mapping. *ACM SIGGRAPH Computer Graphics*, 18(3):207–212, 1984. 3

[7] A. Culotta, T. Kristjansson, A. McCallum, and P. Viola. Corrective feedback and persistent learning for information extraction. *Artificial Intelligence*, 170(14-15):1101–1122, 2006. 1

[8] A. Culotta, A. McCallum, and M. U. A. D. O. C. SCIENCE. Reducing labeling effort for structured prediction tasks. In *PROCEEDINGS OF THE NATIONAL CONFERENCE ON ARTIFICIAL INTELLI-GENCE*, volume 20, page 746. Menlo Park, CA; Cambridge, MA; London; AAAI Press; MIT Press; 1999, 2005. 1

[9] N. Dalal and B. Triggs. Histograms of oriented gradients for human detection. In *CVPR*, pages I: 886–893, 2005. 2

[10] R. Fan, K. Chang, C. Hsieh, X. Wang, and C. Lin. LIBLINEAR: A library for large linear classification. *The Journal of Machine Learning Research*, 9:1871–1874, 2008. 2

[11] P. Felzenszwalb and D. Huttenlocher. Distance transforms of sampled functions. *Cornell Computing and Information Science Technical Report TR2004-1963*, 2004. 3

[12] C. Liu, J. Yuen, A. Torralba, J. Sivic, and W. Freeman. Sift flow: dense correspondence across different scenes. In *Proceedings of the 10th European Conference on Computer Vision: Part III*, pages 28–42. Springer-Verlag, 2008. 1

[13] S. Oh, A. Hoogs, A. Perera, N. Cuntoor, C.-C. Chen, J. T. Lee, S. Mukherjee, J. K. Aggarwal, H. Lee, L. Davis, E. Swears, X. Wang, Q. Ji, K. Reddy, M. Shah, C. Vondrick, H. Pirsiavash, D. Ramanan, J. Yuen, A. Torralba, B. Song, A. Fong, A. Roy-Chowdhury, and M. Desai. A large-scale benchmark dataset for event recognition in surveillance video. In *CVPR*, 2011. 1, 7, 8

[14] B. Settles. Active learning literature survey. Computer Sciences Technical Report 1648, University of Wisconsin–Madison, 2009. 1, 2, 3

[15] C. Vondrick, D. Ramanan, and D. Patterson. Efficiently Scaling Up Video Annotation on Crowdsourced Marketplaces. ECCV, 2010. 1, 2, 8

[16] J. Yuen, B. Russell, C. Liu, and A. Torralba. LabelMe video: Building a Video Database with Human Annotations. 2009. 1, 2

[17] J. Yuen and A. Torralba. A data-driven approach for event prediction. *Computer Vision–ECCV 2010*, pages 707–720, 2010. 1
